# Cluster Kernels for Semi-Supervised Learning

**Olivier Chapelle, Jason Weston, Bernhard Schölkopf**
Max Planck Institute for Biological Cybernetics, 72076 Tübingen, Germany
{*first.last*}*@tuebingen.mpg.de*

## Abstract

We propose a framework to incorporate unlabeled data in kernel classifier, based on the idea that two points in the same cluster are more likely to have the same label. This is achieved by modifying the eigenspectrum of the kernel matrix. Experimental results assess the validity of this approach.

## 1 Introduction

We consider the problem of semi-supervised learning, where one has usually few labeled examples and a lot of unlabeled examples. One of the first semi-supervised algorithms [1] was applied to web page classification. This is a typical example where the number of unlabeled examples can be made as large as possible since there are billions of web page, but labeling is expensive since it requires human intervention. Since then, there has been a lot of interest for this paradigm in the machine learning community; an extensive review of existing techniques can be found in [10].

It has been shown experimentally that under certain conditions, the decision function can be estimated more accurately, yielding lower generalization error [1, 4, 6]. However, in a discriminative framework, it is not obvious to determine how unlabeled data or even the perfect knowledge of the input distribution $P(\mathbf{x})$ can help in the estimation of the decision function. Without any assumption, it turns out that this information is actually useless [10].

Thus, to make use of unlabeled data, one needs to formulate assumptions. One which is made, explicitly or implicitly, by most of the semi-supervised learning algorithms is the so-called "cluster assumption" saying that two points are likely to have the same class label if there is a path connecting them passing through regions of high density only. Another way of stating this assumption is to say that the decision boundary should lie in regions of low density. In real world problems, this makes sense: let us consider handwritten digit recognition and suppose one tries to classify digits 0 from 1. The probability of having a digit which in between a 0 and 1 is very low.

In this article, we will show how to design kernels which implement the cluster assumption, i.e. kernels such that the induced distance is small for points in the same cluster and larger for points in different clusters.

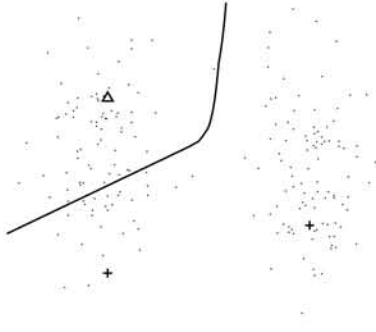

Figure 1: Decision function obtained by an SVM with the kernel (1). On this toy problem, this kernel implements perfectly the cluster assumption: the decision function cuts a cluster only when necessary.

## 2 Kernels implementing the cluster assumption

In this section, we explore different ideas on how to build kernels which take into account the fact that the data is clustered. In section 3, we will propose a framework which unifies the methods proposed in [11] and [5].

### 2.1 Kernels from mixture models

It is possible to design directly a kernel taking into account the generative model learned from the unlabeled data. Seeger [9] derived such a kernel in a Bayesian setting. He proposes to use the unlabeled data to learn a mixture of models and he introduces the *Mutual Information* kernel which is defined in such way that two points belonging to different components of the mixture model will have a low dot product. Thus, in the case of a mixture of Gaussians, this kernel is an implementation of the cluster assumption. Note that in the case of a single mixture model, the *Fisher kernel* [3] is an approximation of this Mutual Information kernel.

Independently, another extension of the Fisher kernel has been proposed in [12] which leads, in the case of a mixture of Gaussians $(\mu_k, \Sigma_k)$ to the *Marginalized kernel* whose behavior is similar to the mutual information kernel,

$$K(\mathbf{x}, \mathbf{y}) = \sum_{k=1}^{q} P(k|\mathbf{x}) P(k|\mathbf{y}) \mathbf{x}^\top \Sigma_k^{-1} \mathbf{y}. \tag{1}$$

To understand the behavior of the Marginalized kernel, we designed a 2D-toy problem (figure 1): 200 unlabeled points have been sampled from a mixture of two Gaussians, whose parameters have then been learned with EM applied to these points. An SVM has been trained on 3 labeled points using the Marginalized kernel (1). The behavior of this decision function is intuitively very satisfying: on the one hand, when not enough label data is available, it takes into account the cluster assumption and does not cut clusters (right cluster), but on the other hand, the kernel is flexible enough to cope with different labels in the same cluster (left side).

### 2.2 Random walk kernel

The kernels presented in the previous section have the drawback of depending on a generative model: first, they require an unsupervised learning step, but more

importantly, in a lot of real world problems, they cannot model the input distribution with sufficient accuracy. When applying the mixture of Gaussians method (presented above) to real world problems, one cannot expect the "ideal" result of figure 1.

For this reason, in clustering and semi-supervised learning, there has been a lot of interest to find algorithms which do not depend on a generative model. We will present two of them, find out how they are related and present a kernel which extends them. The first one is the random walk representation proposed in [11]. The main idea is to compute the RBF kernel matrix (with the labeled and unlabeled points) $K_{ij} = \exp(-\|\mathbf{x}_i - \mathbf{x}_j\|^2 / 2\sigma^2)$ and to interpret it as a transition matrix of a random walk on a graph with vertices $\mathbf{x}_i$, $P(\mathbf{x}_i \to \mathbf{x}_j) = \frac{K_{ij}}{\sum_p K_{ip}}$. After $t$ steps (where $t$ is a parameter to be determined), the probability of going from a point $\mathbf{x}_i$ to a point $\mathbf{x}_j$ should be quite high if both points belong to the same cluster and should stay low if they are in two different clusters.

Let $D$ be the diagonal matrix whose elements are $D_{ii} = \sum_j K_{ij}$. The one step transition matrix is $D^{-1}K$ and after $t$ steps it is $P^t = (D^{-1}K)^t$. In [11], the authors design a classifier which uses directly those transition probabilities. One would be tempted to use $P^t$ as a kernel matrix for a SVM classifier. However, it is not possible to directly use $P^t$ as a kernel matrix since it is not even symmetric. We will see in section 3 how a modified version of $P^t$ can be used as a kernel.

### 2.3   Kernel induced by a clustered representation

Another idea to implement the cluster assumption is to change the representation of the input points such that points in the same cluster are grouped together in the new representation. For this purpose, one can use tools of spectral clustering (see [13] for a review) Using the first eigenvectors of a similarity matrix, a representation where the points are naturally well clustered has been recently presented in [5]. We suggest to train a discriminative learning algorithm in this representation. This algorithm, which resembles kernel PCA, is the following:

1. Compute the *affinity* matrix, which is an RBF kernel matrix but with diagonal elements being 0 instead of 1.

2. Let $D$ be a diagonal matrix with diagonal elements equal to the sum of the rows (or the columns) of $K$ and construct the matrix $L = D^{-1/2}KD^{-1/2}$.

3. Find the eigenvectors $(\mathbf{v}_1, \ldots, \mathbf{v}_k)$ of $L$ corresponding the first $k$ eigenvalues.

4. The new representation of the point $\mathbf{x}_i$ is $(v_{i1}, \ldots, v_{ik})$ and is normalized to have length one: $\varphi(\mathbf{x}_i)_p = v_{ip} / (\sum_{j=1}^k v_{ij}^2)^{1/2}$.

The reason to consider the first eigenvectors of the affinity matrix is the following. Suppose there are $k$ clusters in the dataset infinitely far apart from each other. One can show that in this case, the first $k$ eigenvalues of the affinity matrix will be 1 and the eigenvalue $k+1$ will be strictly less than 1 [5]. The value of this gap depends on how well connected each cluster is: the better connected, the larger the gap is (the smaller the $k+$1st eigenvalue). Also, in the new representation in $\mathbb{R}^k$ there will be $k$ vectors $\mathbf{z}_1, \ldots, \mathbf{z}_k$ orthonormal to each other such that each training point is mapped to one of those $k$ points depending on the cluster it belongs to.

This simple example show that in this new representation points are naturally clustered and we suggest to train a linear classifier on the mapped points.

# 3    Extension of the cluster kernel

Based on the ideas of the previous section, we propose the following algorithm:

1. As before, compute the RBF matrix $K$ from both labeled and unlabeled points (this time with 1 on the diagonal and not 0) and $D$, the diagonal matrix whose elements are the sum of the rows of $K$.

2. Compute $L = D^{-1/2}KD^{-1/2}$ and its eigendecomposition $L = U\Lambda U^{\top}$.

3. Given a transfer function $\varphi$, let $\tilde{\lambda}_i = \varphi(\lambda_i)$, where the $\lambda_i$ are the eigenvalues of $L$, and construct $\tilde{L} = U\tilde{\Lambda}U^{\top}$.

4. Let $\tilde{D}$ be a diagonal matrix with $\tilde{D}_{ii} = 1/\tilde{L}_{ii}$ and compute $\tilde{K} = \tilde{D}^{1/2}\tilde{L}\tilde{D}^{1/2}$.

The new kernel matrix is $\tilde{K}$. Different transfer function lead to different kernels:

**Linear** $\varphi(\lambda) = \lambda$. In this case $\tilde{L} = L$ and $\tilde{D} = D$ (since the diagonal elements of $K$ are 1). It turns out that $\tilde{K} = K$ and no transformation is performed.

**Step** $\varphi(\lambda) = 1$ if $\lambda \geq \lambda_{cut}$ and 0 otherwise. If $\lambda_{cut}$ is chosen to be equal to the $k$-th largest eigenvalue of $L$, then the new kernel matrix $\tilde{K}$ is the dot product matrix in the representation of [5] described in the previous section.

**Linear-step** Same as the step function, but with $\varphi(\lambda) = \lambda$ for $\lambda \geq \lambda_{cut}$. This is closely related to the approach consisting in building a linear classifier in the space given by the first Kernel PCA components [8]: if the normalization matrix $D$ and $\tilde{D}$ were equal to the identity, both approaches would be identical. Indeed, if the eigendecomposition of $K$ is $K = U\Lambda U^{\top}$, the coordinates of the training points in the kernel PCA representation are given by the matrix $U\Lambda^{1/2}$.

**Polynomial** $\varphi(\lambda) = \lambda^t$. In this case, $\tilde{L} = L^t$ and $\tilde{K} = \tilde{D}^{1/2}D^{1/2}(D^{-1}K)^t D^{-1/2}\tilde{D}^{1/2}$. The matrix $D^{-1}K$ is the transition matrix in the random walk described in section 2.2 and $\tilde{K}$ can be interpreted as a normalized and symmetrized version of the transition matrix corresponding to a $t$ step random walk.

This makes the connection between the idea of the random walk kernel of section 2.2 and a linear classifier trained in a space induced by either the spectral clustering algorithm of [5] or the Kernel PCA algorithm.

**How to handle test points**    If test points are available during training and if they are also drawn from the same distribution as the training points (an assumption which is commonly made), then they should be considered as unlabeled points and the matrix $\tilde{K}$ described above should be built using training, unlabeled and test points.

However, it might happen that test points are not available during training. This is a problem, since our method produces a new kernel matrix, but not an analytic form of the effective new kernel that could readily be evaluated on novel test points. In this case, we propose the following solution: approximate a test point $\mathbf{x}$ as a linear combination of the training and unlabeled points, and use this approximation to express the required dot product between the test point and other points in the feature space. More precisely, let

$$\boldsymbol{\alpha}^0 = \arg\min_{\boldsymbol{\alpha}} \left\| \Phi(\mathbf{x}) - \sum_{i=1}^{n+n_u} \alpha_i \Phi(\mathbf{x}_i) \right\| = K^{-1}\mathbf{v}$$

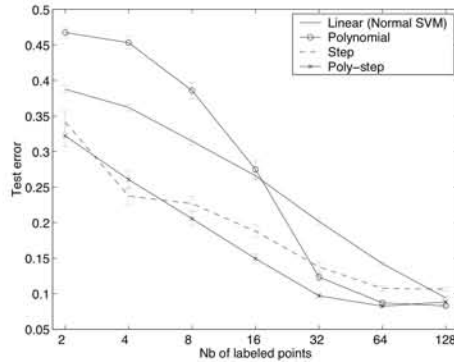

Figure 2: Test error on a text classification problem for training set size varying from 2 to 128 examples. The different kernels correspond to different kind of transfer functions.

with $v_i = K(\mathbf{x}, \mathbf{x}_i)$[1]. Here, $\Phi$ is the feature map corresponding to $K$, i.e., $K(\mathbf{x}, \mathbf{x}') = (\Phi(\mathbf{x}) \cdot \Phi(\mathbf{x}'))$. The new dot product between the test point $\mathbf{x}$ and the other points is expressed as a linear combination of the dot products of $\tilde{K}$,

$$\tilde{K}(\mathbf{x}, \mathbf{x}_i) \equiv (\tilde{K}\boldsymbol{\alpha}^0)_i = (\tilde{K}K^{-1}\mathbf{v})_i.$$

Note that for a linear transfer function, $\tilde{K} = K$, and the new dot product is the standard one.

## 4 Experiments

### 4.1 Influence of the transfer function

We applied the different kernel clusters of section 3 to the text classification task of [11], following the same experimental protocol. There are two categories `mac` and `windows` with respectively 958 and 961 examples of dimension 7511. The width of the RBF kernel was chosen as in [11] giving $\sigma = 0.55$. Out of all examples, 987 were taken away to form the test set. Out of the remaining points, 2 to 128 were randomly selected to be labeled and the other points remained unlabeled. Results are presented in figure 2 and averaged over 100 random selections of the labeled examples. The following transfer functions were compared: linear (i.e. standard SVM), polynomial $\varphi(\lambda) = \lambda^5$, step keeping only the $n + 10$ where $n$ is the number of labeled points, and poly-step defined in the following way (with $1 \geq \lambda_1 \geq \lambda_2 \geq \dots$),

$$\varphi(\lambda_i) = \left\{ \begin{array}{ll} \sqrt{\lambda_i} & i \leq n + 10 \\ \lambda_i^2 & i > n + 10 \end{array} \right.$$

For large sizes of the (labeled) training set, all approaches give similar results. The interesting case are small training sets. Here, the step and poly-step functions work very well. The polynomial transfer function does not give good results for very small training sets (but nevertheless outperforms the standard SVM for medium sizes). This might be due to the fact that in this example, the second largest eigenvalue is 0.073 (the largest is by construction 1). Since the polynomial transfer function tends

to push to 0 the small eigenvalues, it turns out that the new kernel has "rank almost one" and it is more difficult to learn with such a kernel. To avoid this problem, the authors of [11] consider a sparse affinity matrix with non-zeros entries only for neighbor examples. In this way the data are by construction more clustered and the eigenvalues are larger. We verified experimentally that the polynomial transfer function gave better results when applied to a sparse affinity matrix.

Concerning the step transfer function, the value of the cut-off index corresponds to the number of dimensions in the feature space induced by the kernel, since the latter is linear in the representation given by the eigendecomposition of the affinity matrix. Intuitively, it makes sense to have the number of dimensions increase with the number of training examples, that is the reason why we chose a cutoff index equal to $n + 10$.

The poly-step transfer function is somewhat similar to the step function, but is not as rough: the square root tends to put more importance on dimensions corresponding to large eigenvalues (recall that they are smaller than 1) and the square function tends to discard components with small eigenvalues. This method achieves the best results.

## 4.2  Automatic selection of the transfer function

The choice of the poly-step transfer function in the previous choice corresponds to the intuition that more emphasis should be put on the dimensions corresponding to the largest eigenvalues (they are useful for cluster discrimination) and less on the dimensions with small eigenvalues (corresponding to intra-cluster directions). The general form of this transfer function is

$$\varphi(\lambda_i) = \left\{ \begin{array}{ll} \lambda_i^p & i \leq r \\ \lambda_i^q & i > r \end{array} \right. , \tag{2}$$

where $p, q \in \mathbb{R}$ and $r \in \mathbb{N}$ are 3 hyperparameters. As before, it is possible to choose qualitatively some values for these parameters, but ideally, one would like a method which automatically chooses good values. It is possible to do so by gradient descent on an estimate of the generalization error [2]. To assess the possibility of estimating accurately the test error associated with the poly-step kernel, we computed the span estimate [2] in the same setting as in the previous section. We fixed $p = q = 2$ and the number of training points to 16 (8 per class). The span estimate and the test error are plotted on the left side of figure 3.

Another possibility would be to explore methods that take into account the spectrum of the kernel matrix in order to predict the test error [7].

## 4.3  Comparison with other algorithms

We summarized the test errors (averaged over 100 trials) of different algorithms trained on 16 labeled examples in the following table.

| Normal SVM | Transductive SVM [4] | Random walk [11] | Cluster kernel |
|---|---|---|---|
| 27.5% (± 7) | 15.6% (± 2.5) | 15.5% | 12.6% (± 5.3) |

The transductive SVM algorithm consists in maximizing the margin on both labeled and unlabeled. To some extent it implements also the cluster assumption since it tends to put the decision function in low density regions. This algorithm has been successfully applied to text categorization [4] and is a state-of-the-art algorithm for

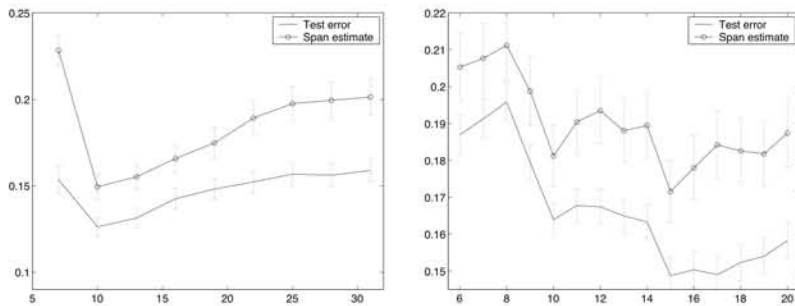

Figure 3: The span estimate predicts accurately the minimum of the test error for different values of the cutoff index $r$ in the **poly-step** kernel (2). Left: text classification task, right: handwritten digit classification

performing semi-supervised learning. The result of the **Random walk** kernel is taken directly from [11]. Finally, the cluster kernel performance has been obtained with $p = q = 2$ and $r = 10$ in the transfer function 2. The value of $r$ was the one minimizing the span estimate (see left side of figure 3).

Future experiments include for instance the Marginalized kernel (1) with the standard generative model used in text classification by Naive Bayes classifier [6].

## 4.4  Digit recognition

In a second set of experiments, we considered the task of classifying the handwritten digits 0 to 4 against 5 to 9 of the USPS database. The cluster assumption should apply fairly well on this database since the different digits are likely to be clustered.

2000 training examples have been selected and divided into 50 subsets on 40 examples. For a given run, one of the subsets was used as the labeled training set, whereas the other points remained unlabeled. The width of the RBF kernel was set to 5 (it was the value minimizing the test error in the supervised case).

The mean test error for the standard SVM is 17.8% (standard deviation 3.5%), whereas the transductive SVM algorithm of [4] did not yield a significant improvement (17.6% ± 3.2%). As for the cluster kernel (2), the cutoff index $r$ was again selected by minimizing the span estimate (see right side of figure 3). It gave a test error of 14.9% (standard deviation 3.3%). It is interesting to note in figure 3 the local minimum at $r = 10$, which can be interpreted easily since it corresponds to the number of different digits in the database.

It is somehow surprising that the transductive SVM algorithm did not improve the test error on this classification problem, whereas it did for text classification. We conjecture the following explanation: the transductive SVM is more sensitive to outliers in the unlabeled set than the cluster kernel methods since it directly tries to maximize the margin on the unlabeled points. For instance, in the top middle part of figure 1, there is an unlabeled point which would have probably perturbed this algorithm. However, in high dimensional problems such as text classification, the influence of outlier points is smaller. Another explanation is that this method can get stuck in local minima, but that again, in higher dimensional space, it is easier to get out of local minima.

# 5 Conclusion

In a discriminative setting, a reasonable way to incorporate unlabeled data is through the cluster assumption. Based on the ideas of spectral clustering and random walks, we proposed a framework for constructing kernels which implement the cluster assumption: the induced distance depends on whether the points are in the same cluster or not. This is done by changing the spectrum of the kernel matrix. Since there exist several bounds for SVMs which depend on the shape of this spectrum, the main direction for future research is to perform automatic model selection based on these theoretical results. Finally, note that the cluster assumption might also be useful in a purely supervised learning task.

**Acknowledgments**

The authors would like to thank Martin Szummer for helpful discussion on this topic and for having provided us with his database.

## Footnotes

[1]We consider here an RBF kernel and for this reason the matrix $K$ is always invertible.

# References

[1] A. Blum and T. Mitchell. Combining labeled and unlabeled data with co-training. In *COLT: Proceedings of the Workshop on Computational Learning Theory*. Morgan Kaufmann Publishers, 1998.

[2] O. Chapelle, V. Vapnik, O. Bousquet, and S. Mukherjee. Choosing multiple parameters for support vector machines. *Machine Learning*, 46(1-3):131–159, 2002.

[3] T. Jaakkola and D. Haussler. Exploiting generative models in discriminative classifiers. In *Advances in Neural Information Processing*, volume 11, pages 487–493. The MIT Press, 1998.

[4] T. Joachims. Transductive inference for text classification using support vector machines. In *Proceedings of the 16th International Conference on Machine Learning*, pages 200–209. Morgan Kaufmann, San Francisco, CA, 1999.

[5] A. Y. Ng, M. I. Jordan, and Y. Weiss. On spectral clustering: Analysis and an algorithm. In *Advances in Neural Information Processing Systems*, volume 14, 2001.

[6] K. Nigam, A. K. McCallum, S. Thrun, and T. M. Mitchell. Learning to classify text from labeled and unlabeled documents. In *Proceedings of AAAI-98, 15th Conference of the American Association for Artificial Intelligence*, pages 792–799, Madison, US, 1998. AAAI Press, Menlo Park, US.

[7] B. Schölkopf, J. Shawe-Taylor, A. J. Smola, and R. C. Williamson. Generalization bounds via eigenvalues of the Gram matrix. Technical Report 99-035, NeuroColt, 1999.

[8] B. Schölkopf, A. Smola, and K.-R. Müller. Nonlinear component analysis as a kernel eigenvalue problem. *Neural Computation*, 10:1299–1310, 1998.

[9] M. Seeger. Covariance kernels from Bayesian generative models. In *Advances in Neural Information Processing Systems*, volume 14, 2001.

[10] M. Seeger. Learning with labeled and unlabeled data. Technical report, Edinburgh University, 2001.

[11] M. Szummer and T. Jaakkola. Partially labeled classification with markov random walks. In *Advances in Neural Information Processing Systems*, volume 14, 2001.

[12] K. Tsuda, T. Kin, and K. Asai. Marginalized kernels for biological sequences. *Bioinformatics*, 2002. To appear. Also presented at ICMB 2002.

[13] Y. Weiss. Segmentation using eigenvectors: A unifying view. In *International Conference on Computer Vision*, pages 975–982, 1999.
